# Sphere Embedding:
# An Application to Part-of-Speech Induction

**Yariv Maron**
Gonda Brain Research Center
Bar-Ilan University
Ramat-Gan 52900, Israel
*syarivm@yahoo.com*

**Michael Lamar**
Department of Mathematics and Computer Science
Saint Louis University
St. Louis, MO 63103, USA
*mlamar@slu.edu*

**Elie Bienenstock**
Division of Applied Mathematics
And Department of Neuroscience
Brown University
Providence, RI 02912, USA
*elie@brown.edu*

## Abstract

Motivated by an application to unsupervised part-of-speech tagging, we present an algorithm for the Euclidean embedding of large sets of categorical data based on co-occurrence statistics. We use the CODE model of Globerson et al. but constrain the embedding to lie on a high-dimensional unit sphere. This constraint allows for efficient optimization, even in the case of large datasets and high embedding dimensionality. Using $k$-means clustering of the embedded data, our approach efficiently produces state-of-the-art results. We analyze the reasons why the sphere constraint is beneficial in this application, and conjecture that these reasons might apply quite generally to other large-scale tasks.

## 1    Introduction

The embedding of objects in a low-dimensional Euclidean space is a form of dimensionality reduction that has been used in the past mostly to create 2D representations of data for the purpose of visualization and exploratory data analysis [10, 13]. Most methods work on objects of a single type, endowed with a measure of similarity. Other methods, such as [3], embed objects of heterogeneous types, based on their co-occurrence statistics. In this paper we demonstrate that the latter can be successfully applied to unsupervised part-of-speech (POS) induction, an extensively studied, challenging, problem in natural language processing [1, 4, 5, 6, 7].

The problem we address is *distributional* POS tagging, in which words are to be tagged based on the statistics of their immediate left and right context in a corpus (ignoring morphology and other features). The induction task is fully unsupervised, *i.e.*, it uses no annotations. This task has been addressed in the past using a variety of methods. Some approaches, such as [1], combine a Markovian assumption with clustering. Many recent works use HMMs, perhaps due to their excellent performance on the supervised version of the task [7, 2, 5]. Using a latent-descriptor clustering approach, [15] obtain the best results to date for distributional-only unsupervised POS tagging of the widely-used WSJ corpus.

Using a heterogeneous-data embedding approach for this task, we define separate embedding

functions for the objects "left word" and "right word" based on their co-occurrence statistics, *i.e.*, based on bigram frequencies. We are interested in modeling the statistical interactions between left words and right words, as relevant to POS tagging, rather than their joint distribution. Indeed, modeling the joint distribution directly results in models that do not handle rare words well. We use the CODE (Co-Occurrence Data Embedding) model of [3], where statistical interaction is modeled as the negative exponential of the Euclidean distance between the embedded points. This embedding model incorporates the marginal probabilities, or unigram frequencies, in a way that results in appropriate handling of both frequent and rare words.

The size of the dataset (number of points to embed) and the embedding dimensionality are several-fold larger than in the applications studied in [3], making the optimization methods used by these authors impractical. Instead, we use a simple and intuitive stochastic-gradient procedure. Importantly, in order to handle both the large dataset and the relatively high dimensionality of the embedding needed for this application, we constrain the embedding to lie on the unit sphere. We therefore refer to this method as *Spherical CODE*, or **S-CODE**. The spherical constraint causes the regularization term—the partition function—to be nearly constant and also makes the stochastic gradient ascent smoother; this allows a several-fold computational improvement, and yields excellent performance. After convergence of the embedding model, we use a *k*-means algorithm to cluster all the words of the corpus, based on their embeddings. The induced POS labels are evaluated using the standard setting for this task, yielding state-of-the-art tagging performance.

## 2 Methods
### 2.1 Model

We represent a bigram, *i.e.*, an ordered pair of adjacent words in the corpus, as joint random variables $(X,Y)$, each taking values in $W$, the set of word types occurring in the corpus. Since $X$ and $Y$, the first and second words in a bigram, play different roles, we build a *heterogeneous* model, *i.e.*, use two embedding functions, $\phi(x)$ and $\psi(y)$. Both map $W$ into $S$, the unit sphere in the *r*-dimensional Euclidean space.

We use $\bar{p}$ for the word-type frequencies: $\bar{p}(x)$ is the number of word tokens of type $x$ divided by the total number of tokens in the corpus. We refer to $\bar{p}$ as the empirical marginal distribution, or unigram frequency. We use $\bar{p}(x,y)$ for the empirical joint distribution of $X$ and $Y$, *i.e.*, the distribution of bigrams $(X,Y)$. Because our ultimate goal is the clustering of word types for POS tagging, we want the embedding to be insensitive to the marginals: two word types with similar context distributions should be mapped to neighboring points in $S$ even if their unigram frequencies are very different. We therefore use the *marginal-marginal* model of [3], defined by:

$$p(x,y) = \tfrac{1}{Z}\bar{p}(x)\bar{p}(y)e^{-d_{x,y}^2} \tag{1}$$

$$d_{x,y}^2 = \|\phi(x) - \psi(y)\|_{L_2}^2 = \sum_{i=1}^{r}\big(\phi_i(x) - \psi_i(y)\big)^2 \tag{2}$$

$$Z = \sum_{x,y}\bar{p}(x)\bar{p}(y)e^{-d_{x,y}^2} \tag{3}$$

The log-likelihood, $\lambda$, of the corpus of bigrams is the expected value, under the empirical bigram distribution, of the log of the model bigram probability:

$$\lambda = \sum_{x,y}\bar{p}(x,y)\,log\,p(x,y) = -\sum_{x,y}\bar{p}(x,y)d_{x,y}^2 - log\,Z + \sum_{x,y}\bar{p}(x,y)log\,\bar{p}(x)\bar{p}(y) \tag{4}$$

The model is parameterized by $2 \times |W|$ points on the unit sphere $S$ in $r$ dimensions: $\{\phi(x)\}_{x \in W}$ and $\{\psi(y)\}_{y \in W}$. These points are initialized randomly, *i.e.*, independently and uniformly on $S$.

To maximize the likelihood, we use a gradient-ascent approach. The gradient of the log likelihood is as follows (observe that the last term in (4) does not depend on the model, hence does not contribute to the gradient):

$$\frac{\partial \lambda}{\partial \phi(u)} = \sum_y 2\bar{p}(u,y)[\psi(y) - \phi(u)] + \frac{1}{Z}\sum_y 2\bar{p}(u)\bar{p}(y)[\phi(u) - \psi(y)]e^{-d_{u,y}^2} \quad (5)$$

$$\frac{\partial \lambda}{\partial \psi(v)} = \sum_x 2\bar{p}(x,v)[\phi(x) - \psi(v)] + \frac{1}{Z}\sum_x 2\bar{p}(x)\bar{p}(v)[\psi(v) - \phi(x)]e^{-d_{x,v}^2} \quad (6)$$

For sufficiently large problems such as POS tagging of a large corpus, computing the partition function, $Z$, after each gradient step or even once every fixed number of steps can be impractical. Instead, it turns out (see Discussion) that, thanks to the sphere constraint, we can approximate this dynamic variable, $Z$, using a constant, $\tilde{Z}$, which arises from a coarse approximation in which all pairs of embedded variables are distributed uniformly and independently on the sphere. Thus, we set $(\phi(X), \psi(Y)) \sim (U_1, U_2)$ with $U_1$ and $U_2$ *i.i.d.* uniformly on $S$, and get our estimate $\tilde{Z}$ as the expected value of the resulting random variable, $e^{-\|U_1 - U_2\|^2}$:

$$\tilde{Z} = E\left[e^{-\|U_1 - U_2\|^2}\right]. \quad (7)$$

Numerical evaluation of (7) yields $\tilde{Z} \approx 0.146$ for the 25-dimensional sphere. An even coarser approximation can be obtained by noting that, for large $r$, the random variable $\|U_1 - U_2\|^2 = 2 - 2U_1 \cdot U_2$ is fairly peaked around 2 (the random variable $U_1 \cdot U_2$ is close to a Student's $t$ with $r$ degrees of freedom, compressed by a factor of $\sqrt{r}$). This yields the estimate $\tilde{Z} \approx e^{-2} \approx 0.135$.

For the present application, we find that performance does not suffer from using a constant $\tilde{Z}$ rather than recomputing $Z$ often during gradient-ascent. It is also fairly robust to the choice of $\tilde{Z}$. We observe only minor changes in performance for $\tilde{Z}$ ranging over [0.1, 0.5].

We use sampling to compute a stochastic approximation of the gradient. To implement the first sum in (5) and (6) − representing an attraction force between the embeddings of the words in a bigram − we sample bigrams from the empirical joint $\bar{p}(x, y)$. Given a sample $(x_1, y_1)$, only the $\phi(x_1)$ and $\psi(y_1)$ parameter vectors are updated. The partial updates that emerge from these two sums are:

$$\phi(x_1) := \phi(x_1) + \eta[\psi(y_1) - \phi(x_1)] \quad (8)$$
$$\psi(y_1) := \psi(y_1) + \eta[\phi(x_1) - \psi(y_1)], \quad (9)$$

where $\eta$ is the step size. In order to speed up the convergence process, we use a learning rate that decreases as word types are repeatedly observed. If $C(w)$ is the number of times word type $w$ has been previously encountered, we use:

$$\eta(C(w)) = \eta_0\left(\frac{\varphi_0}{\varphi_0 + C(w)}\right), \quad \varphi_0 = 100, \ \eta_0 = 0.1. \quad (10)$$

The model is very robust to the choice of the function $\eta(C)$, as long as it decreases smoothly. This modified learning rate also reduces the variability of the tagging accuracy, while slightly increasing its mean.

The second sum in (5) and in (6) − representing a repulsion force − involves not the empirical joint but the product of the empirical marginals. Thus, the complete update is:

$$\phi(x_1) := \phi(x_1) + \eta(C(x_1)) \left[ [\psi(y_1) - \phi(x_1)] + \frac{e^{-d_{x_1,y_2}^2}}{\bar{z}} [\phi(x_1) - \psi(y_2)] \right] \qquad (11)$$

$$\psi(y_1) := \psi(y_1) + \eta(C(y_1)) \left[ [\phi(x_1) - \psi(y_1)] + \frac{e^{-d_{x_2,y_1}^2}}{\bar{z}} [\psi(y_1) - \phi(x_2)] \right], \qquad (12)$$

where $(x_1, y_1)$ is sampled from the joint $\bar{p}(x,y)$, and $x_2$ and $y_2$ are sampled from the marginal $\bar{p}(x)$ independently from each other and independently from $x_1$ and $y_1$. After each step, the updated vectors are projected back onto the sphere $S$.

After convergence, for any word $w$, we have two embedded vectors, $\phi(w)$ and $\psi(w)$. These vectors are concatenated to form a single geometric description of word type $w$. The collection of all these vectors is then clustered using a weighted $k$-means clustering algorithm: in each iteration, a cluster's centroid is updated as the *weighted* mean of its currently assigned constituent vectors, with the weight of the vector for word $w$ equal to $\bar{p}(w)$. The number of clusters chosen depends on whether evaluation is to be done against the PTB45 or the PTB17 tagset (see below, Section 2.2).[1]

## 2.2    Evaluation and data

The resulting assignment of cluster labels to word types is used to label the corpus. The standard practice for evaluating the performance of the induced labels is to either map them to the gold-standard tags, or to use an information-theoretic measure. We use the three evaluation criteria that are most common in the recent literature. The first criterion maps each cluster to the POS tag that it best matches according to the hand-annotated labels. The match is determined by finding the tag that is most frequently assigned to any token of any word type in the cluster. Because the criterion is free to assign several clusters to the same POS tag, this evaluation technique is called many-to-one mapping, or MTO. Once the map is constructed, the accuracy score is obtained as the fraction of all tokens whose inferred tag under the map matches the hand-annotated tag.

The second criterion, 1-to-1 mapping, is similar to the first, but the mapping is restricted from assigning multiple clusters to a single tag; hence it is called one-to-one mapping, or 1-to-1. Most authors construct the 1-to-1 mapping greedily, assigning maximal-score label-to-tag matches first; some authors, e.g. [15], use the optimal map. Once the map is constructed, the accuracy is computed just as in MTO. The third criterion, variation of information, or VI, is a map-free information-theoretic metric [9, 2].

We note that we and other authors found the most reliable criterion for comparing unsupervised POS taggers to be MTO. However, we include all three criteria for completeness.

We use the Wall Street Journal part of the Penn Treebank [8] (1,173,766 tokens). We ignore capitalization, leaving 43,766 word types, to compare performance with other models consistently. Evaluation is done against the full tag set (PTB45), and against a coarse tag set (PTB17) [12]. For PTB45 evaluation, we use either 45 or 50 clusters, in order for our results to be comparable to all recent works. For PTB17 evaluation, we use 17 clusters, as do all other authors.

## 3    Results

Figure 1 shows the model performance when evaluated with several measures. MTO17 and MTO50 refer to the number of tokens tagged correctly under the many-to-1 mapping for the PTB45 and PTB17 tagsets respectively. The type-accuracy curves use the same mapping

and tagsets, but record the fraction of word types whose inferred tag matches their "modal" annotated tag, *i.e.*, the annotated tag co-occurring most frequently with this word type. We also show the scaled log likelihood, to illustrate its convergence. These results were produced using a constant, pre-computed, $\tilde{Z}$. Using this constant value allows the model to run in a matter of minutes rather than the hours or days required by HMMs and MRFs.

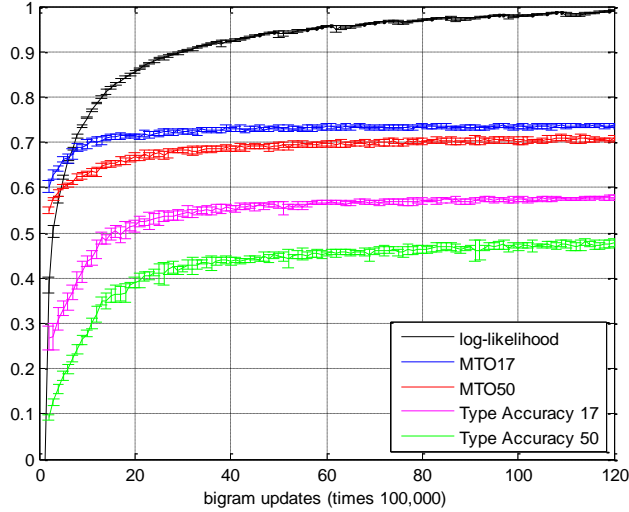

Figure 1: Scores against number of iterations (bigram updates). Scores are averaged over 10 sessions, and shown with 1-std error bars. MTO17 is the Many-to-1 tagging accuracy score based on 17 induced labels mapped to 17 tags. MTO50 is the Many-to-1 score based on 50 induced labels mapped to 45 tags. Type Accuracy 17 (50) is the average accuracy per word type, where the gold-standard tag of a word type is the *modal* annotated tag of that type (see text). All runs used $\tilde{Z} = 0.154$, $r=25$.

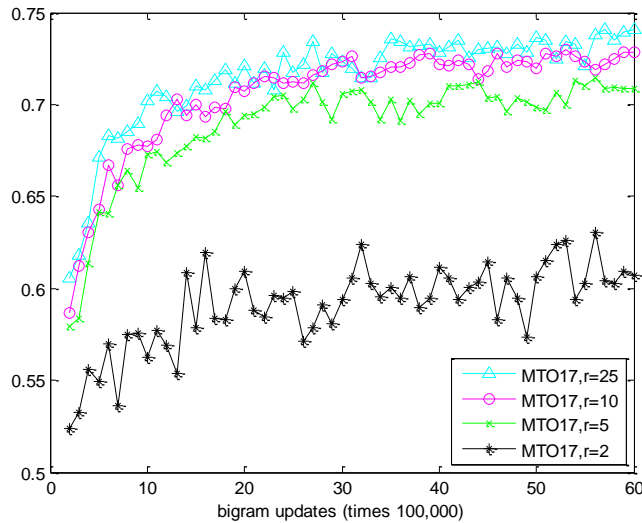

Figure 2: Comparison of models with different dimensionalities: $r = 2, 5, 10, 25$. MTO17 is the Many-to-1 score based on 17 induced labels mapped to PTB17 tags.

Figure 2 shows the model performance for different dimensionalities $r$. As $r$ increases, so does the performance. Unlike previous applications of CODE [3] (which often emphasize

visualization of data and thus require a low dimension), this unsupervised POS-tagging application benefits from high values of $r$. Larger values of $r$ cause both the tagging accuracy to improve and the variability during convergence to decrease.

| Model | Many-to-1 | | | 1-to-1 | | | VI | | |
|---|---|---|---|---|---|---|---|---|---|
| | PTB17 | PTB45-45 | PTB45-50 | PTB17 | PTB45-45 | PTB45-50 | PTB17 | PTB45-45 | PTB45-50 |
| S-CODE (Z=0.1456) | 73.8 (0.5) | **68.8** (0.16) | **70.4** (0.5) | 52.2 | 50.0 | 50.0 | 2.93 | 3.46 | 3.46 |
| S-CODE (Z=0.3) | 74.5 (0.2) | **68.6** (0.16) | **71.5** (0.6) | 54.9 | 48.7 | 48.8 | 2.80 | 3.38 | 3.39 |
| LDC | **75.1** (0.04) | 68.1 (0.2) | **71.2** (0.06) | 59.3 | | 48.3 | | | |
| Brown | | 67.8 | **70.5** | | 50.1 | 51.3 | | 3.47 | 3.45 |
| HMM-EM | 64.7 | | 62.1 | 43.1 | | 40.5 | 3.86 | | 4.48 |
| HMM-VB | 63.7 | | 60.5 | 51.4 | | 46.1 | 3.44 | | 4.28 |
| HMM-GS | 67.4 | | 66.0 | 44.6 | | 49.9 | 3.46 | | 4.04 |
| HMM-Sparse(32) | 70.2 | 65.4 | | 49.5 | 44.5 | | | | |
| VEM $(10^{-1},10^{-1})$ | 68.2 | 54.6 | | 52.8 | 46.0 | | | | |

Table 1: Comparison to other models, under three different evaluation measures. S-CODE uses $r = 25$ dimensions. It was run 10 times, each with $12 \cdot 10^6$ update steps. LDC is from [15]; Brown shows the best results from [14] and website mentioned therein; HMM-EM, HMM-VB and HMM-GS show the best results from [2]; HMM-Sparse(32) and VEM show the best results from [5]. The numbers in parentheses are standard deviations. For the VI criterion, lower values are better. PTB45-45 maps 45 induced labels to 45 tags, while PTB45-50 maps 50 induced labels to 45 tags.

Table 1 compares our model, S-CODE, to previous state-of-the-art approaches. Under the Many-to-1 criterion, which we find to be the most appropriate of the three for the evaluation of unsupervised POS taggers, S-CODE is superior to HMM results, and scores comparably to [15], the highest-performing model to date on this task.

We find that the model is very robust to the choice of $\tilde{Z}$ within the range 0.1 to 0.5. This robustness lends promise for the usefulness of this method for other applications in which the partition function is impractical to compute. This point is discussed further in the next section.

## 4    Discussion

The problem of embedding heterogeneous categorical data $(X,Y)$ based on their co-occurrence statistics may be formulated as the task of finding a pair of maps $\phi(x)$ and $\psi(y)$ such that, for any pair $(x,y)$, the distance between the images of $x$ and $y$ reflects the statistical interaction between them. Such embeddings have been used mostly for the purpose of visualization and exploratory data analysis. Here we demonstrate that embedding can be successfully applied to a well-studied computational-linguistics task, achieving state-of-the-art performance.

### 4.1    S-CODE v. CODE

The approach proposed here, S-CODE, is a variant of the CODE model of [3]. In the task at hand, the sets $X$ and $Y$ to be embedded are large (43K), making most conventional

embedding approaches, including CODE (as implemented in [3]), impractical. As explained below, S-CODE overcomes the large-dataset challenge by constraining the maps to lie on the unit sphere. It uses stochastic gradient ascent to maximize the likelihood of the model.

The gradient of the log-likelihood w.r.t. a given $\phi(x)$ includes two components, each with a simple intuitive meaning. The first component embodies an attraction force, pulling $\phi(x)$ toward $\psi(y)$ in proportion to the empirical joint $\bar{p}(x, y)$. The second component, the gradient of the regularization term, $-\log Z$, embodies a repulsion force; it keeps the solution away from the trivial state where all $x$'s and $y$'s are mapped to the same point, and more generally attempts to keep $Z$ small. The repulsion force pushes $\phi(x)$ away from $\psi(y)$ in proportion to the product of the empirical marginals $\bar{p}(x)$ and $\bar{p}(y)$, and is scaled by $e^{-d_{x,y}^2}/Z$. The computational complexity of $Z$, the partition function, is $|X| \times |Y| \times r$.

In the application studied here, the use of the spherical constraint of S-CODE has two important consequences. First, it makes the computation of $Z$ unnecessary. Indeed, when using the spherical constraint, we observed that $Z$, when actually computed and updated every $10^6$ steps, does not deviate much from its initial value. For example, for $r = 25$, $Z$ rises smoothly from 0.145 to 0.182. Note that the absolute minimum of $Z$—obtained for a $\phi$ that maps all of $W$ to a single point on $S$ and a $\psi$ that maps all of $W$ to the opposite point—is $e^{-4}$; the absolute maximum of $Z$, obtained for $\phi$ and $\psi$ that map all of $W$ to the same point, is 1. We also observed that replacing $Z$, in the update algorithm, by any constant $\tilde{Z}$ in the range [.1 .5] does not dramatically alter the behavior of the model. We nevertheless note that larger values of $\tilde{Z}$ tend to yield a slightly higher performance of the POS tagger built from the model. Note that the only effect of changing $\tilde{Z}$ in the stochastic gradient algorithm is to change the relative strength of the attraction and repulsion terms.

We compared the performance of S-CODE with CODE. The original CODE implementation [3] could not support the size of our data set. To overcome this limitation, we used the stochastic-gradient method described above, but without projecting to the sphere. This required us to compute the partition function, which is highly computationally intensive. We therefore computed the partition function only once every $q$ update steps (where one update step is the sampling of one bigram). We found that for $q = 10^5$ the partition function and likelihood changed smoothly enough and converged, and the embeddings yielded tagging performances that did not differ significantly from those obtained with S-CODE. The second important consequence of imposing the spherical constraint is that it makes the stochastic gradient-ascent procedure markedly smoother. As a result, a relatively large step size can be used, achieving convergence and excellent tagging performance in about 10 minutes of computation time on a desktop machine. CODE requires a smaller step size as well as the recomputation of the partition function, and, as a result, computation time in this application was 6 times longer than with S-CODE.

When gauging the applicability of S-CODE to different large-scale embedding problems, one should try to gain some understanding of why the spherical constraint stabilizes the partition function, and whether $Z$ will stabilize around the same value for other problems. The answer to the first question appears to be that the regularization term is not so strong as to prevent clusters from forming—this is demonstrated by the excellent performance of the model when used for POS tagging—yet it is strong enough to enforce a fairly uniform distribution of these clusters on the sphere—resulting in a fairly stable value of $Z$. One may reasonably conjecture that this behavior will generalize to other problems. To answer the second question, we note that the order of magnitude of $Z$ is essentially set by the coarsest of the two estimates derived in Section 2, namely $e^{-2} \approx 0.135$, and that this estimate is problem-independent. As a result, S-CODE is, in principle, applicable to datasets of much larger size than the present problem. The computational complexity of the algorithm is $O(Nr)$, and the memory requirement is $O(|W|r)$ where $N$ is the number of word tokens, and $|W|$ is the number of word types. In contrast, and as mentioned above, CODE, even in our stochastic-gradient version, is considerably more computationally intensive; it would clearly be completely impractical for much larger datasets.

## 4.2    Comparison to other POS induction models

Even though embedding models have been studied extensively, they are not widely used for

POS tagging (see however [18]). For the unsupervised POS tagging task, HMMs have until recently dominated the field. Here we show that an embedding model substantially outperforms HMMs, and achieves the same level of performance as the best distributional-only model to date [15]. Models that use features, e.g. morphological, achieve higher tagging precision [11, 14]. Incorporating features into S-CODE can easily be done, either directly or in a two-step approach as in [14]; this is left for future work.

One of the widely-acknowledged challenges in applying HMMs to the unsupervised POS tagging problem is that these models do not afford a convenient vehicle to modeling an important sparseness property of natural languages, namely the fact that any given word type admits of only a small number of POS tags—often only one (see in particular [7, 2, 4]). In contrast, the approach presented here maps each word type to a single point in $\mathbb{R}^r$. Hence, it assigns a single tag to each word type, like a number of other recent approaches [15, 16, 17]. These approaches are incapable of disambiguating, *i.e.*, of assigning different tags to the same word depending on context, as in "I long to see a long movie." HMMs are, in principle, capable of doing so, but at the cost of over-parameterization. In view of the superior performance of S-CODE and of other type-level approaches, it appears that under-parameterization might be the better choice for this task.

Another difference between our model and HMMs previously applied to this problem is that our model is symmetric, thereby modeling right *and* left context distributions. In contrast, HMMs are asymmetric in that they typically model a left-to-right transition and would find a different solution if a right-to-left transition were modeled. We argue that using both distributions in a symmetric way better captures the important linguistic information. In the past, left and right distributions were extracted by factoring the bigram matrix and using the left and right eigenvectors. Such a linear method does not handle rare words well. Instead, we choose to learn the ratio $p(x,y)/(\bar{p}(x)\bar{p}(y))$. This approach allows words with similar contexts but different unigram frequencies to be embedded near each other.

Like HMMs, CODE provides a model of the distribution of the data at hand. S-CODE departs slightly from this framework. Since it does not use the exact partition function in the stochastic gradient ascent procedure—and was actually found to perform best when replacing $Z$, in the update rule, by a constant that is substantially larger than the true value of $Z$—it only approximately converges to a local maximum of a likelihood function. In future work, and as a more radical deviation from the CODE model, one may then give up altogether modeling the distribution of $X$ and $Y$, instead relying on a heuristically motivated objective function of sphere-constrained embeddings $\phi(x)$ and $\psi(y)$, to be maximized. Preliminary studies using a number of alternative functional forms for the regularization term yielded promising results.

Although S-CODE and LDC [15] achieve essentially the same level of performance on taggings that induce 17, 45, or 50 labels (Table 1), S-CODE proves superior for the induction of very fine-grained taggings. Thus, we compared the performances of S-CODE and LDC on the task of inducing 300 labels. Under the MTO criterion, LDC achieved 80.9% (PTB45) and 87.9% (PTB17). S-CODE significantly outperformed it, with 83.5% (PTB45) and 89.8% (PTB17).

The appeal of S-CODE lies not only in its strong performance on the unsupervised POS tagging problem, but also in its simplicity, its robustness, and its mathematical grounding. The mathematics underlying CODE, as developed in [3], are intuitive and relatively simple. Modeling the joint probability of word type co-occurrence through distances between Euclidean embeddings, without relying on discrete categories or states, is a novel and promising approach for POS tagging. The spherical constraint introduced here permits the approximation of the partition function by a constant, which is the key to the efficiency of the algorithm for large datasets. The stochastic-gradient procedure produces two competing forces with intuitive meaning, familiar from the literature on learning in generative models. While the accuracy and computational efficiency of S-CODE is matched by the recent LDC algorithm [15], S-CODE is more robust, showing very little change in performance over a wide range of implementation choices. We expect that this improved robustness will allow S-CODE to be easily and successfully applied to other large-scale tasks, both linguistic and non-linguistic.

## Footnotes

[1] Source code is available at the author's website: *faculty.biu.ac.il/~marony*.

# References

[1] Alexander Clark. 2003. Combining distributional and morphological information for part of speech induction. In *10th Conference of the European Chapter of the Association for Computational Linguistics*, pages 59–66.

[2] Jianfeng Gao and Mark Johnson. 2008. A comparison of bayesian estimators for unsupervised Hidden Markov Model POS taggers. In *Proceedings of the 2008 Conference on Empirical Methods in Natural Language Processing*, pages 344–352.

[3] Amir Globerson, Gal Chechik, Fernando Pereira, and Naftali Tishby. 2007. Euclidean embedding of co-occurrence data. *Journal of Machine Learning Research,* 8:2265–2295.

[4] Sharon Goldwater and Tom Griffiths. 2007. A fully Bayesian approach to unsupervised part-of-speech tagging. In *Proceedings of the 45th Annual Meeting of the Association of Computational Linguistics*, pages 744–751.

[5] João V. Graça, Kuzman Ganchev, Ben Taskar, and Fernando Pereira. 2009. Posterior vs. Parameter Sparsity in Latent Variable Models. In *Neural Information Processing Systems Conference (NIPS)*.

[6] Aria Haghighi and Dan Klein. 2006. Prototype-driven learning for sequence models. In *Proceedings of the Human Language Technology Conference of the NAACL, Main Conference*, pages 320–327.

[7] Mark Johnson. 2007. Why doesn't EM find good HMM POS-taggers? In *Proceedings of the 2007 Joint Conference on Empirical Methods in Natural Language Processing and Computational Natural Language Learning (EMNLP-CoNLL)*, pages 296–305.

[8] M.P. Marcus, M.A. Marcinkiewicz, and B. Santorini. 1993. Building a large annotated corpus of English: The Penn Treebank. *Computational linguistics*, 19(2):313–330.

[9] Marina Meilă. 2003. Comparing clusterings by the variation of information. In Bernhard Schölkopf and Manfred K. Warmuth, editors, *COLT 2003: The Sixteenth Annual Conference on Learning Theory*, volume 2777 of *Lecture Notes in Computer Science*, pages 173–187. Springer.

[10] Sam T. Roweis and Lawrence K. Saul. 2000. Nonlinear dimensionality reduction by locally linear embedding. *Science*, 290:2323–2326.

[11] Taylor Berg-Kirkpatrick, Alexandre Bouchard-Côté, John DeNero, and Dan Klein. 2010. Painless Unsupervised Learning with Features. In *Human Language Technologies: The 2010 Annual Conference of the North American Chapter of the Association for Computational Linguistics*, pages 582-590.

[12] Noah A. Smith and Jason Eisner. 2005. Contrastive estimation: Training log-linear models on unlabeled data. In *Proceedings of the 43rd Annual Meeting of the Association for Computational Linguistics (ACL'05)*, pages 354–362.

[13] Joshua B. Tenenbaum, Vin de Silva, and John C. Langford. 2000. A global geometric framework for nonlinear dimensionality reduction. *Science*, 290:2319–2323.

[14] Christos Christodoulopoulos, Sharon Goldwater and Mark Steedman. 2010. Two Decades of Unsupervised POS induction: How far have we come? In *Proceedings of the 2010 Conference on Empirical Methods in Natural Language Processing (EMNLP 2010)*, pages 575–584.

[15] Michael Lamar, Yariv Maron and Elie Bienenstock. 2010. Latent-Descriptor Clustering for Unsupervised POS Induction. In *Proceedings of the 2010 Conference on Empirical Methods in Natural Language Processing,* pages 799–809.

[16] Yoong Keok Lee, Aria Haghighi, and Regina Barzilay. 2010. Simple Type-Level Unsupervised POS Tagging. In *Proceedings of the 2010 Conference on Empirical Methods in Natural Language Processing*, pages 853-861.

[17] Michael Lamar, Yariv Maron, Mark Johnson, Elie Bienenstock. 2010. SVD and clustering for unsupervised POS tagging. In *Proceedings of the ACL 2010 Conference Short Papers*, pages 215-219.

[18] Ronan Collobert and Jason Weston. 2008. A unified architecture for natural language processing: Deep neural networks with multitask learning. In *Proceedings of the Twenty-fifth International Conference on Machine Learning (ICML 2008)*, pages 160–167.
